# Using matrices to model symbolic relationships

**Ilya Sutskever and Geoffrey Hinton**
University of Toronto
{ilya, hinton}@cs.utoronto.ca

## Abstract

We describe a way of learning matrix representations of objects and relationships. The goal of learning is to allow multiplication of matrices to represent symbolic relationships between objects and symbolic relationships between relationships, which is the main novelty of the method. We demonstrate that this leads to excellent generalization in two different domains: modular arithmetic and family relationships. We show that the same system can learn first-order propositions such as $(2, 5) \in +3$ or *(Christopher, Penelope) ∈ has_wife*, and higher-order propositions such as $(3, +3) \in$ *plus* and $(+3, -3) \in$ *inverse* or *(has_husband, has_wife) ∈ higher_oppsex*. We further demonstrate that the system understands how higher-order propositions are related to first-order ones by showing that it can correctly answer questions about first-order propositions involving the relations $+3$ or *has_wife* even though it has not been trained on any first-order examples involving these relations.

## 1 Introduction

It is sometimes possible to find a way of mapping objects in a "data" domain into objects in a "target" domain so that operations in the data domain can be modelled by operations in the target domain. If, for example, we map each positive number to its logarithm, multiplication in the data domain can be modelled by addition in the target domain. When the objects in the data and target domains are more complicated than single numbers, it may be difficult to find good mappings using inspiration alone. If we consider a continuous space of possible mappings and if we define a smooth measure of how well any particular mapping works, it is possible to use gradient search to find good mappings between the data and target domains.

Paccanaro and Hinton [10] introduced a method called "Linear Relational Embedding" (LRE) that uses multiplication of vectors by matrices in the target domain to model pairwise relations between objects in the data domain. LRE applies to a finite set of objects $\Omega$ and a finite set of relations $\mathcal{R}$ where every relation $R \in \mathcal{R}$ is a set of pairs of objects, so $R \subseteq \Omega \times \Omega$. Given the objects and relations, LRE finds a column-vector representation $\mathbf{A}$ of each object $A \in \Omega$, and a matrix representation $\mathbf{R}$ of each relation $R \in \mathcal{R}$, such that the product $\mathbf{RA}$ is close to $\mathbf{B}$ for all pairs $(A, B)$ that are members of the relation $R$, and far from $\mathbf{C}$ for all pairs $(A, C)$ that are not members of $R$. LRE learns the vectors and matrices by performing gradient descent in a cost function $\mathcal{C}$ that measures the similarities between $\mathbf{RA}$ and all $\mathbf{B}$ such that $(A, B) \in R$ relative to the similarities between $\mathbf{RA}$ and the vector representations of all the objects in the set of known objects $\Omega$:

$$\mathcal{C} = -\sum_R \sum_{(A,B)\in R} \log \frac{\exp(-\|\mathbf{RA} - \mathbf{B}\|^2)}{\sum_{C\in\Omega} \exp(-\|\mathbf{RA} - \mathbf{C}\|^2)} \tag{1}$$

The cost function in Eq. 1 is "discriminative" because it compares the distance from $\mathbf{RA}$ to each correct answer with the distances from $\mathbf{RA}$ to all possible answers. This prevents trivial solutions

in which $\mathbf{RA}$ and $\mathbf{B}$ are always zero, but it also causes the cost function to be nonconvex, making it hard to optimize. We can view $\exp(-\|\mathbf{RA} - \mathbf{B}\|^2)$ as the unnormalized probability density of $\mathbf{B}$ under a spherical Gaussian centered at $\mathbf{RA}$. The cost function then represents the sum of the negative log probabilities of picking the correct answers to questions of the form $(A,?) \in R$ if we pick answers stochastically in proportion to their probability densities under the spherical Gaussian centered at $\mathbf{RA}$.

We say that LRE accurately models a set of objects and relations if its answers to queries of the form $(A, ?) \in R$ are correct, which means that for each object $A$ and relation $R$ such that there are $k$ objects $X$ satisfying $(A, X) \in R$, each vector representation $\mathbf{X}$ of each such object $X$ must be among the $k$ closest vector representations to $\mathbf{RA}$. The definition of correctness implies that LRE's answer to a query $(A, ?) \in R$ that has no solutions is always trivially correct. More refined versions of LRE handle such unsatisfiable queries more explicitly [9].

It may not be obvious how to determine if the representation found by LRE is good. One way is to check if LRE's representation generalizes to test data. More specifically, if LRE has not been informed that $B$ is an answer to the query $(A, ?) \in R$ that has $k$ correct answers (that is, $(A, B)$ was removed from $R$ during LRE's learning), yet LRE answers the query $(A, ?) \in R$ correctly by placing $\mathbf{B}$ among the $k$ closest object representations to $\mathbf{RA}$, then we can claim that LRE's representation generalizes. Such generalization can occur only if LRE learned the "right" representations $\mathbf{A}, \mathbf{B}$, and $\mathbf{R}$ from the other propositions, which can happen only if the true relation is plausible according to LRE's inductive bias that determines the subjective plausibility of every possible set of objects and relations (see, e.g., [6]). If the representation is high-dimensional, then LRE can easily represent any set of relations that is not too large, so its inductive bias finds all sets of relations plausible, which prevents generalization from being good. However, if the representation is low-dimensional, then LRE must make use of regularities in the training set in order to accurately model the data, but if it succeeds in doing so, generalization will be good. Paccanaro and Hinton [10] show that low-dimensional LRE exhibits excellent generalization on datasets such as the family relations task. In general, the dimensionality of the representation should grow with the total numbers of objects and relations, because when there are few objects and relations, a high-dimensional representation easily overfits, but if the number of objects and relations is large then the dimensionality can be higher, without overfitting. The best dimensionality depends on the "fit" between LRE and the data, and is mainly an empirical question.

A drawback of LRE is that the square matrices it uses to represent relations are quadratically more cumbersome than the vectors it uses to represent objects. This causes the number of free parameters to grow rapidly when the dimensionality of the representations is increased. More importantly, it also means that relations cannot themselves be treated as objects. Paccanaro and Hinton [10], for example, describe a system that learns propositions of the form: $(2, 5) \in +3$ where $+3$ is a relation that is represented by a learned matrix, but their system does not understand that the learned matrix for $+3$ has anything in common with the learned vector that is used to model the number 3 in propositions like $(5, 3) \in -2$.

In this paper we describe "Matrix Relational Embedding" (MRE), which is a version of LRE that uses matrices as the representation for objects as well as for relations.[1] MRE optimizes the same cost function as LRE (equation 1), with the difference that $\mathbf{RA} - \mathbf{C}$ is now a matrix rather than a vector and $\|\mathbf{RA} - \mathbf{C}\|^2$ denotes the sum of the squares of the entries of the matrix. This choice of matrix norm makes MRE a direct generalization of LRE. All distances between matrices will be computed using this norm.

Although MRE is a simple variation of LRE, it has two important advantages.

The first advantage of MRE is that when using an $N \times N$ matrix to represent each object it is possible to make $N$ much smaller than when using an $N$-dimensional vector, so MRE can use about the same number of parameters as LRE for each object but many fewer parameters than LRE for each relation, which is useful for "simple" relations.

The second advantage of MRE, which is also the main novelty of this paper, is that MRE is capable of representing higher-order relations, instances of which are $(+3, -3) \in inverse$ or $(has\_husband, has\_wife) \in higher\_oppsex$. It can also represent relations involving an object and a relation, for instance $(3, +3) \in plus$. Formally, we are given a finite set of higher-order relations $\tilde{\mathcal{R}}$, where a higher-order relation $\tilde{R} \in \tilde{\mathcal{R}}$ is a relation whose arguments can be relations as well as objects, which we formalize as $\tilde{R} \subseteq \mathcal{R} \times \mathcal{R}$ or $\tilde{R} \subseteq \Omega \times \mathcal{R}$ ($\mathcal{R}$ is the set of the basic relations). The matrix representation of MRE allows it to treat relations in (almost) the same way it treats basic objects, so there is no difficulty representing relations whose arguments are also relations.

We show that MRE can answer questions of the form $(4, ?) \in +3$ even though the training set contains no examples of the basic relation $+3$. It can do this because it is told what $+3$ means by being given higher-order information about $+3$. It is told that $(3, +3) \in plus$ and it figures out what $plus$ means from higher-order examples of the form $(2, +2) \in plus$ and basic examples of the form $(3, 5) \in +2$. This enables MRE to understand a relation from an "analogical definition": if it is told that $has\_father$ to $has\_mother$ is like $has\_brother$ to $has\_sister$, etc., then MRE can answer queries involving $has\_father$ based on this analogical information alone. Finally, we show that MRE can learn new relations after an initial set of objects and relations has already been learned and the learned matrices have been fixed. This shows that MRE can add new knowledge to previously acquired propositions without the need to relearn the original propositions. We believe that MRE is the first gradient-descent learning system that can learn new relations from definitions, including learning the meanings of the terms used in the definitions. This significantly extends the symbolic learning abilities of connectionist-type learning algorithms.

Some of the existing connectionist models for representing and learning relations and analogies [2, 4] are able to detect new relations and to represent hierarchical relations of high complexity. They differ by using temporal synchrony for explicitly representing the binding of the relations to object, and, more importantly, do not use distributed representations for representing the relations themselves.

## 2    The modular arithmetic task

Paccanaro and Hinton [10] describe a very simple modular arithmetic task in which the 10 objects are the numbers from 0 to 9 and the 9 relations are +0 to +4 and −1 to −4. Linear Relational Embedding easily learns this task using two-dimensional vectors for the numbers and $2 \times 2$ matrices for the relations. It arranges the numbers in a circle centered at the origin and uses rotation matrices to implement the relations. We used base 12 modular arithmetic, thus there are 12 objects, and made the task much more difficult by using both the twelve relations +0 to +11 and the twelve relations ×0 to ×11. We did not include subtraction and division because in modular arithmetic every proposition involving subtraction or division is equivalent to one involving addition or multiplication.

There are 288 propositions in the modular arithmetic ntask. We tried matrices of various sizes and discovered that $4 \times 4$ matrices gave the best generalization when some of the cases are held-out. We held-out 30, 60, or 90 test cases chosen at random and used the remaining cases to learn the real-valued entries of the 12 matrices that represent numbers and the 24 matrices that represent relations. The learning was performed by gradient descent in the cost function in Eq. 1. We repeated this five times with a different random selection of held-out cases each time. Table 1 shows the number of errors on the held-out test cases.

## 3    Details of the learning procedure

To learn the parameters, we used the conjugate gradient optimization algorithm available in the "scipy" library of the Python programming language with the default optimization parameters. We computed the gradient of the cost function on all of the training cases before updating the parameters, and initialized the parameters by a random sample from a spherical Gaussian with unit variance on each dimension. We also included "weight-decay" by adding $0.01 \sum_i w_i^2$ to the cost function, where $i$ indexes all of the entries in the matrices for objects and relations. The variance of the results is due to the nonconvexity of the objective function. The implementation is available in [www.cs.utoronto.ca/~ilya/code/2008/mre.tar.gz].

Test results for the basic modular arithmetic.

|  | errors on 5 test sets |  |  |  |  | mean test error |
|---|---|---|---|---|---|---|
| (30) | 0 | 0 | 0 | 0 | 0 | 0.0 |
| (60) | 29 | 4 | 0 | 1 | 0 | 6.8 |
| (90) | 27 | 23 | 16 | 31 | 23 | 24.0 |

Table 1: Test results on the basic modular arithmetic task. Each entry shows the number of errors on the randomly held-out cases. There were no errors on the training set. Each test query has 12 possible answers of which 1 is correct, so random guessing should be incorrect on at least 90% of the test cases. The number of held-out cases of each run is written in brackets.

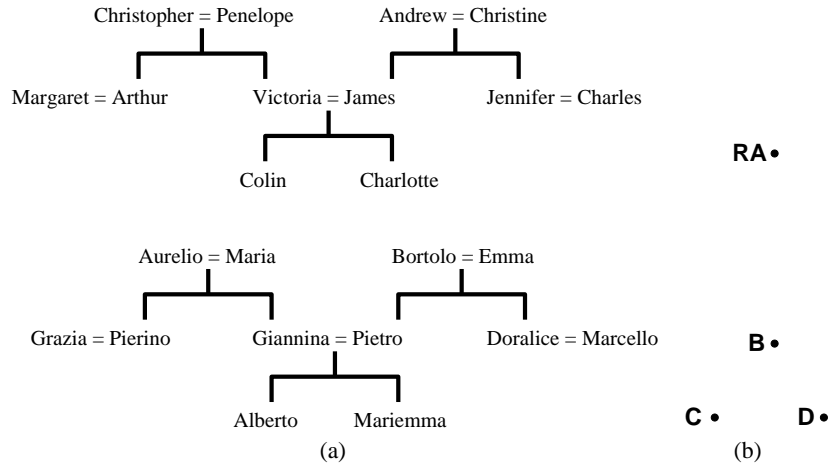

(a)                          (b)

Figure 1: (a) Two isomorphic family trees (b) An example of a situation in which the discriminative cost function in Eq. 1 causes the matrix **RA** produced by MRE to be far from the correct answer, **B** (see section 5).

In an attempt to improve generalization, we tried constraining all of the $4 \times 4$ matrices by setting half of the elements of each matrix to zero so that they were each equivalent to two independent $2 \times 2$ matrices. Separate experiments showed that $2 \times 2$ matrices were sufficient for learning either the mod 3 or the mod 4 version of our modular arithmetic task, so the mod 12 version can clearly be done using a pair of $2 \times 2$ matrices for each number or relation. However, the gradient optimization gets stuck in poor local minima.

## 4    The standard family trees task

The "standard" family trees task defined in [3] consists of the two family trees shown in figure 1(a) where the relations are {*has_husband, has_wife, has_son, has_daughter, has_father, has_mother, has_brother, has_sister, has_nephew, has_niece, has_uncle, has_aunt*}. Notice that for the last four relations there are people in the families in figure 1(a) for whom there are two different correct answers to the question $(A,?) \in R$. When there are $N$ correct answers, the best way to maximize the sum of the log probabilities of picking the correct answer on each of the $N$ cases is to produce an output matrix that is equidistant from the $N$ correct answers and far from all other answers. If the designated correct answer on such a case is not among the $N$ closest, we treat that case as an error. If we count cases with two correct answers as two different cases the family trees task has 112 cases.

We used precisely the same learning procedure and weight-decay as for the modular arithmetic task. We held-out 10, 20, or 30 randomly selected cases as test cases, and we repeated the random selection of the test cases five times. Table 2 shows the number of errors on the test cases when $4 \times 4$ matrices are learned for each person and for each relation. MRE generalizes much better than the

Test results for the basic family trees task.

| | errors on 5 test sets | | | | | mean test error |
|---|---|---|---|---|---|---|
| (10) | 0 | 0 | 0 | 0 | 2 | 0.4 |
| (20) | 6 | 0 | 0 | 0 | 0 | 1.2 |
| (30) | 0 | 2 | 4 | 0 | 4 | 2.0 |

Table 2: Test results on the basic family trees task. Each entry shows the number of errors on the randomly held-out cases. There were no errors on the training set. The same randomly selected test sets were used for the $4 \times 4$ matrices. Each test query has 24 possible answers, of which at most 2 objects are considered correct. As there are 24 objects, random guessing is incorrect on at least 90% of the cases.

feedforward neural network used by [3] which typically gets one or two test cases wrong even when only four test cases are held-out. It also generalizes much better than all of the many variations of the learning algorithms used by [8] for the family trees task. These variations cannot achieve zero test errors even when only four test cases are held-out and the cases are chosen to facilitate generalization.

## 5 The higher-order modular arithmetic task

We used a version of the modular arithmetic task in which the only basic relations were $\{+0, +2, \ldots, +11\}$, but we also included the higher-order relations *plus, minus, inverse* consisting of 36 propositions, examples of which are $(3, +3) \in plus$; $(3, +9) \in minus$; $(+3, +9) \in inverse$. We then held-out all of the examples of one of the basic relations and trained $4 \times 4$ matrices on all of the other basic relations plus all of the higher-order relations.

Our first attempt to demonstrate that MRE could generalize from higher-order relations to basic relations failed: the generalization was only slightly better than chance. The failure was caused by a counter-intuitive property of the discriminative objective function in Eq. 1 [9]. When learning the higher-order training case $(3, +3) \in plus$ it is not necessary for the product of the matrix representing 3 and the matrix representing *plus* to be exactly equal to the matrix representing $+3$. The product only needs to be closer to $+3$ than to any of the other matrices. In cases like the one shown in figure 1(b), the *relative* probability of the point $\mathbf{B}$ under a Gaussian centered at $\mathbf{RA}$ is increased by moving $\mathbf{RA}$ up, because this lowers the unnormalized probabilities of $\mathbf{C}$ and $\mathbf{D}$ by a greater proportion than it lowers the unnormalized probability of $\mathbf{B}$. The discriminative objective function prevents all of the representations collapsing to the same point, but it does not force the matrix products to be exactly equal to the correct answer. As a result, the representation of $+3$ produced by the product of 3 and *plus* does not work properly when it is applied to a number.

To overcome this problem, we modified the cost function for training the higher-order relations so that it is minimized when $\tilde{\mathbf{R}}\mathbf{A}$ is exactly equal to $\mathbf{B}$

$$\mathcal{C} = \sum_{\tilde{R} \in \tilde{\mathcal{R}}} \sum_{(A,B) \in \tilde{R}} \|\tilde{\mathbf{R}}\mathbf{A} - \mathbf{B}\|^2, \tag{2}$$

where $\tilde{R}$ ranges over $\tilde{\mathcal{R}}$, the set of all higher-order relations, and $A$ and $B$ can be either relations or basic objects, depending on $\tilde{R}$'s domain.

Even when using this non-discriminative cost function for training the higher-order relations, the matrices could not all collapse to zero because the discriminative cost function was still being used for training the basic relations. With this modification, the training caused the product of 3 and *plus* to be very close to $+3$ and, as a result, there was often good generalization to basic relations even when all of the basic relations involving $+3$ were removed from MRE's training data and all it was told about $+3$ was that $(3, +3) \in plus$, $(9, +3) \in minus$, and $(+9, +3) \in inverse$ (see table 3).

Test results for higher-order arithmetic task.

|  | errors on 5 test sets | | | | | mean test error |
|---|---|---|---|---|---|---|
| +1 (12) | 5 | 0 | 0 | 0 | 0 | 1.0 |
| +4 (12) | 0 | 0 | 6 | 6 | 1 | 2.6 |
| +6 (12) | 0 | 6 | 4 | 4 | 0 | 2.8 |
| +10 (12) | 3 | 8 | 0 | 0 | 7 | 3.6 |

Table 3: Test results on the higher-order arithmetic task. Each row shows the number of incorrectly answered queries involving a relation (i.e., +1, +4, +6, or +10) all of whose basic examples were removed from MRE's training data, so MRE's knowledge of this relation was entirely from the other higher-order relations. Learning was performed 5 times starting from different initial random parameters. There were no errors on the training set for any of the runs. The number of test cases is written in brackets.

Test results for the higher-order family trees task.

|  | errors on 5 test sets | | | | | mean test error |
|---|---|---|---|---|---|---|
| *has_father* (12) | 0 | 12 | 0 | 0 | 0 | 2.4 |
| *has_aunt* (8) | 4 | 8 | 4 | 0 | 4 | 4.0 |
| *has_sister* (6) | 2 | 0 | 0 | 0 | 0 | 0.4 |
| *has_nephew* (8) | 0 | 0 | 8 | 0 | 0 | 1.6 |

Table 4: Test results for the higher-order family trees task. In each row, all basic propositions involving a relation are held-out (i.e., *has_father*, *has_aunt*, *has_sister*, or *has_nephew*). Each row shows the number of errors MRE makes on these held-out propositions on 5 different learning runs from different initial random parameters. The only information MRE has on these relations is in the form of a single higher-order relation, *higher_oppsex*. There were no errors on the training sets for any of the runs. The number of held-out cases is written in brackets.

## 6   The higher-order family trees task

To demonstrate that similar performance is obtained on family trees task when higher-order relations are used, we included in addition to the 112 basic relations the higher-order relation *higher_oppsex*. To define *higher_oppsex* we observe that many relations have natural male and natural female versions, as in: mother-father, nephew-niece, uncle-aunt, brother-sister, husband-wife, and son-daughter. We say that $(A, B) \in higher\_oppsex$ for relations $A$ and $B$ if $A$ and $B$ can be seen as natural counterparts in this sense. Four of the twelve examples of *higher_oppsex* are given below:

1. *(has_father, has_mother)* $\in higher\_oppsex$
2. *(has_mother, has_father)* $\in higher\_oppsex$
3. *(has_brother, has_sister)* $\in higher\_oppsex$
4. *(has_sister, has_brother)* $\in higher\_oppsex$

We performed an analogous test to that in the previous section on the higher order modular arithmetic task, using exactly the same learning procedure and learning parameters. For the results, see table 4.

The family trees task and its higher-order variant may appear difficult for systems such as MRE or LRE because of the logical nature of the task, which is made apparent by hard rules such as $(A, B) \in has\_father$, $(A, C) \in has\_brother \Rightarrow (C, B) \in has\_father$. However, MRE does not perform any explicit logical deduction based on explicitly inferred rules, as would be done in an Inductive Logic Programming system (e.g., [7]). Instead, it "precomputes the answers" to all queries during training, by finding the matrix representation that models its training set. Once the representation is found, many correct facts become "self-evident" and do not require explicit derivation. Humans may be using a somewhat analogous mechanism (thought not necessarily one with matrix multiplications), since when mastering a new and complicated set of concepts, some humans start by relying heavily on relatively explicit reasoning using the definitions. With experience, however, many nontrivial correct facts may become intuitive to such an extent that experts can make true conjectures whose explicit derivation would be long and difficult. New theorems are easily discovered when the representations of all the concepts make the new theorem intuitive and self-evident.

The sequential higher-order arithmetic task.

|  | errors on 5 test sets | | | | | mean test error |
|---|---|---|---|---|---|---|
| +1 (12) | 0 | 0 | 0 | 2 | 4 | 1.2 |
| +4 (12) | 10 | 8 | 8 | 0 | 3 | 5.8 |
| +6 (12) | 0 | 0 | 4 | 9 | 0 | 2.6 |
| +10 (12) | 0 | 4 | 8 | 0 | 10 | 4.4 |

The sequential higher-order family trees task.

|  | errors on 5 test sets | | | | | mean test error |
|---|---|---|---|---|---|---|
| *has_father* (12) | 0 | 0 | 0 | 10 | 0 | 2.0 |
| *has_aunt* (8) | 0 | 0 | 0 | 8 | 0 | 1.6 |
| *has_sister* (6) | 0 | 0 | 0 | 0 | 0 | 0.0 |
| *has_nephew* (8) | 0 | 0 | 0 | 0 | 0 | 0.0 |

Table 5: Test results for the higher-order arithmetic task (top) and the higher-order family trees task (bottom) when a held-out basic relation is learned from higher-order propositions after the rest of the objects and relations have been learned and fixed. There were no errors on the training propositions. Each entry shows the number of test errors, and the number of test cases is written in brackets.

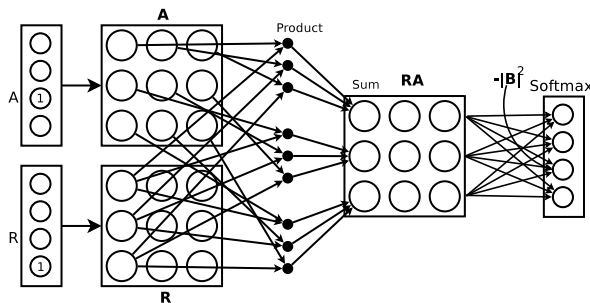

Figure 2: A neural network that is equivalent to Matrix Relational Embedding (see text for details).

This is analogous to the idea that humans can avoid a lot of explicit search when playing chess by "compiling" the results of previous searches into a more complex evaluation function that uses features which make the value of a position immediately obvious.

This does not mean that MRE can deal with general logical data of this kind, because MRE will fail when there are many relations that have many special cases. The special cases will prevent MRE from finding low dimensional matrices that fit the data well and cause it to generalize much more poorly.

## 7  Adding knowledge incrementally

The previous section shows that MRE can learn to apply a basic relation correctly even though the training set only contains higher-order propositions about the relation. We now show that this can be achieved incrementally. After learning some objects, basic relations, and higher-order relations, we freeze the weights in all of the matrices and learn the matrix for a new relation from a few higher-order propositions. Table 5 shows that this works about as well as learning all of the propositions at the same time.

## 8  An equivalent neural network

Consider the neural network shown in Figure 2. The input vectors $R$ and $A$ represent a relation and an object using a one-of-N encoding. If the outgoing weights from the two active input units are set to $\mathbf{R}$ and $\mathbf{A}$, these localist representations are converted into activity patterns in the first hidden layer that represent the matrices $\mathbf{R}$ and $\mathbf{A}$. The central part of the network consists of "sigma-pi" units [12], all of whose incoming and outgoing connections have fixed weights of 1. The sigma-pi units perform a matrix multiplication by first taking the products of pairs of activities in the first hidden layer and then summing the appropriate subsets of these products. As a result, the activities in the next layer represent the matrix $\mathbf{RA}$. The output layer uses a "softmax" function to compute the probability of each possible answer and we now show that if the weights and biases of the output

units are set correctly, this is equivalent to picking answers with a probability that is proportional to their probability density under a spherical Gaussian centered at $\mathbf{RA}$. Consider a particular output unit that represents the answer $B$. If the weights into this unit are set to $2\mathbf{B}$ and its bias is set to $-\|\mathbf{B}\|^2$, the total input to this unit will be:

$$\text{Total input} = -\|\mathbf{B}\|^2 + 2\sum_{ij}(\mathbf{RA})_{ij}\mathbf{B}_{ij} \tag{3}$$

The probability that the softmax assigns to $B$ will therefore be:

$$
\begin{aligned}
p(B|A, R) &= \frac{e^{-\|\mathbf{B}\|^2 + 2\sum_{ij}(\mathbf{RA})_{ij}\mathbf{B}_{ij}}}{\sum_C e^{-\|\mathbf{C}\|^2 + 2\sum_{ij}(\mathbf{RA})_{ij}\mathbf{C}_{ij}}} \\
&= \frac{e^{-\|\mathbf{B}\|^2 + 2\sum_{ij}(\mathbf{RA})_{ij}\mathbf{B}_{ij} - \|\mathbf{RA}\|^2}}{\sum_C e^{-\|\mathbf{C}\|^2 + 2\sum_{ij}(\mathbf{RA})_{ij}\mathbf{C}_{ij} - \|\mathbf{RA}\|^2}} = \frac{e^{-\|\mathbf{RA}-\mathbf{B}\|^2}}{\sum_C e^{-\|\mathbf{RA}-\mathbf{C}\|^2}}
\end{aligned}
\tag{4}
$$

Maximizing the log probability of $p(B|R, A)$ is therefore equivalent to minimizing the cost function given in Eq. 1.

The fact that MRE generalizes much better than a standard feedforward neural network on the family trees task is due to two features. First, it uses the same representational scheme (i.e., the same matrices) for the inputs and the outputs, which the standard net does not; a similar representational scheme was used in [1] to accurately model natural language. Second, it uses "sigma-pi" units that facilitate multiplicative interactions between representations. It is always possible to approximate such interactions in a standard feedforward network, but it is often much better to build them into the model [13, 5, 11].

### Acknowledgments

We would like to thank Alberto Paccanaro and Dafna Shahaf for helpful discussions. This research was supported by NSERC and CFI. GEH holds a Canada Research Chair in Machine Learning and is a fellow of the Canadian Institute for Advanced Research.

## Footnotes

[1] We have also experimented with a version of LRE that learns to *generate* a learned matrix representation of a relation from a learned vector representation of the relation. This too makes it possible to treat relations as objects because they both have vector representations. However, it is less straightforward than simply representing objects by matrices and it does not generalize quite as well.

## References

[1] Y. Bengio, R. Ducharme, P. Vincent, and C. Janvin. A neural probabilistic language model. *The Journal of Machine Learning Research*, 3:1137–1155, 2003.

[2] L.A.A. Doumas, J.E. Hummel, and C.M. Sandhofer. A Theory of the Discovery and Predication of Relational Concepts. *psychological Review*, 115(1):1, 2008.

[3] G.E. Hinton. Learning distributed representations of concepts. *Proceedings of the Eighth Annual Conference of the Cognitive Science Society*, pages 1–12, 1986.

[4] J.E. Hummel and K.J. Holyoak. A Symbolic-Connectionist Theory of Relational Inference and Generalization. *Psychological Review*, 110(2):220–264, 2003.

[5] R. Memisevic and G.E. Hinton. Unsupervised learning of image transformations. *Proceedings of IEEE Conference on Computer Vision and Pattern Recognition*, 2007.

[6] T.M. Mitchell. The need for biases in learning generalizations. *Readings in Machine Learning. Morgan Kaufmann*, 1991.

[7] S. Muggleton and L. De Raedt. Inductive logic programming: Theory and methods. *Journal of Logic Programming*, 19(20):629–679, 1994.

[8] R.C. O'Reilly. *The LEABRA Model of Neural Interactions and Learning in the Neocortex*. PhD thesis, Carnegie Mellon University, 1996.

[9] A. Paccanaro. *Learning Distributed Representations of Relational Data Using Linear Relational Embedding*. PhD thesis, University of Toronto, 2002.

[10] A. Paccanaro and G. Hinton. Learning Distributed Representations of Concepts using Linear Relational Embedding. *IEEE Transactions on Knowledge and Data Engineering*, 13(2):232–245, 2001.

[11] R.P.N. Rao and D.H. Ballard. Development of localized oriented receptive fields by learning a translation-invariant code for natural images. *Network: Computation in Neural Systems*, 9(2):219–234, 1998.

[12] D.E. Rumelhart, G.E. Hinton, and J.L. McClelland. A general framework for parallel distributed processing. *Mit Press Computational Models Of Cognition And Perception Series*, pages 45–76, 1986.

[13] J.B. Tenenbaum and W.T. Freeman. Separating Style and Content with Bilinear Models. *Neural Computation*, 12(6):1247–1283, 2000.
